# Evaluating neuronal codes for inference using Fisher information

**Ralf M. Haefner**[*] **and Matthias Bethge**
Centre for Integrative Neuroscience, University of Tübingen,
Bernstein Center for Computational Neuroscience, Tübingen,
Max Planck Institute for Biological Cybernetics
Spemannstr. 41, 72076 Tübingen, Germany

## Abstract

Many studies have explored the impact of response variability on the quality of sensory codes. The source of this variability is almost always assumed to be intrinsic to the brain. However, when inferring a particular stimulus property, variability associated with other stimulus attributes also effectively act as noise. Here we study the impact of such stimulus-induced response variability for the case of binocular disparity inference. We characterize the response distribution for the binocular energy model in response to random dot stereograms and find it to be very different from the Poisson-like noise usually assumed. We then compute the Fisher information with respect to binocular disparity, present in the monocular inputs to the standard model of early binocular processing, and thereby obtain an upper bound on how much information a model could theoretically extract from them. Then we analyze the information loss incurred by the different ways of combining those inputs to produce a scalar single-neuron response. We find that in the case of depth inference, monocular stimulus variability places a greater limit on the extractable information than intrinsic neuronal noise for typical spike counts. Furthermore, the largest loss of information is incurred by the standard model for position disparity neurons (tuned-excitatory), that are the most ubiquitous in monkey primary visual cortex, while more information from the inputs is preserved in phase-disparity neurons (tuned-near or tuned-far) primarily found in higher cortical regions.

## 1 Introduction

Understanding how the brain performs statistical inference is one of the main problems of theoretical neuroscience. In this paper, we propose to apply the tools developed to evaluate the information content of neuronal codes corrupted by noise to address the question of how well they support statistical inference. At the core of our approach lies the interpretation of neuronal response variability due to nuisance stimulus variability as noise.

Many theoretical and experimental studies have probed the impact of intrinsic response variability on the quality of sensory codes ([1, 12] and references therein). However, most neurons are responsive to more than one stimulus attribute. So when trying to infer a particular stimulus property, the brain needs to be able to ignore the effect of confounding attributes that also influence the neuron's response. We propose to evaluate the usefulness of a population code for inference over a particular parameter by treating the neuronal response variability due to nuisance stimulus attributes as noise equivalent to intrinsic noise (e.g. Poisson spiking).

We explore the implications of this new approach for the model system of stereo vision where the inference task is to extract depth from binocular images. We compute the Fisher information present

---

[*]Corresponding author (ralf.haefner@gmail.com)

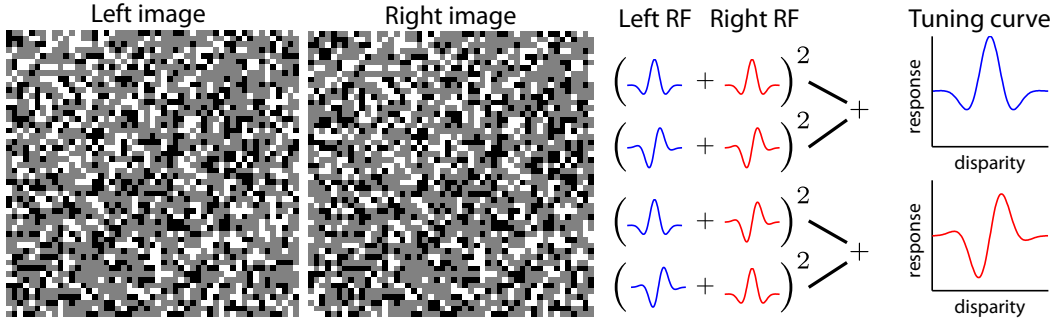

Figure 1: Left: Example random dot stereogram (RDS). Right: Illustration of bincular energy model without (top) and with (bottom) phase disparity.

in the monocular inputs to the standard model of early binocular processing and thereby obtain an upper bound on how precisely a model could theoretically extract depth. We compare this with the amount of information that remains after early visual processing. We distinguish the two principal model flavors that have been proposed to explain the physiological findings. We find that one of the two models appears superior to the other one for inferring depth.

We start by giving a brief introduction to the two principal flavors of the binocular energy model. We then retrace the processing steps and compute the Fisher information with respect to depth inference that is present: first in the monocular inputs, then after binocular combination, and finally for the resulting tuning curves.

## 2 Binocular disparity as a model system

Stereo vision has the advantage of a clear separation between the relevant stimulus dimension – binocular disparity – and the confounding or nuisance stimulus attributes – monocular image structure ([9]). The challenge in inferring disparity in image pairs consists in distinguishing true from false matches, regardless of the monocular structures in the two images. The stimulus that tests this system in the most general way are random dot stereograms (RDS) that consist of nearly identical dot patterns in either eye (see Figure 1). The fact that parts of the images are displaced horizontally with respect to each other has been shown to be sufficient to give rise to a sensation of depth in humans and monkeys ([5, 4]). Since RDS do not contain any monocular depth cues (e.g. size or perspective) the brain needs to correctly match the monocular image features across eyes to compute disparity.

The standard model for binocular processing in primary visual cortex (V1) is the binocular energy model ([5, 10]). It explains the response of disparity-selective V1 neurons by linearly combining the output of monocular simple cells and passing the sum through a squaring nonlinearity (illustrated in Figure 1).

$$r_{\text{even}} = (\nu_L^e + \nu_R^e)^2 + (\nu_L^o + \nu_R^o)^2 = \nu_L^{e\,2} + \nu_L^{o\,2} + \nu_R^{e\,2} + \nu_R^{o\,2} + 2(\nu_L^e \nu_R^e + \nu_L^o \nu_R^o). \qquad (1)$$

where $\nu_L^e$ is the output of an even-symmetric receptive field (RF) applied to the left image, $\nu_R^o$ is the output of an odd-symmetric receptive field (RF) applied to the right image, etc. By pairing an even and an odd-symmetric RF in each eye[1], the monocular part of the response of the cell $\nu_L^{e\,2} + \nu_L^{o\,2} + \nu_R^{e\,2} + \nu_R^{o\,2}$ becomes invariant to the monocular phase of a grating stimulus (since $\sin^2 + \cos^2 = 1$) and the binocular part is modulated only by the difference (or disparity) between the phases in left and right grating – as observed for complex cells in V1. The disparity tuning curve resulting from the combination in equation (1) is even-symmetric (illustrated in Figure 1 in blue) and is one of two primary types of tuning curves found in cortex ([5]). In order to model the other, odd-symmetric type of tuning curves (Figure 1 in red), the filter outputs are combined such that the output of an even-symmetric filter is always combined with that of an odd-symmetric one in the other eye:

$$r_{\text{odd}} = (\nu_L^e + \nu_R^o)^2 + (\nu_L^o + \nu_R^e)^2 = \nu_L^{e\,2} + \nu_L^{o\,2} + \nu_R^{e\,2} + \nu_R^{o\,2} + 2(\nu_L^e \nu_R^o + \nu_L^o \nu_R^e). \qquad (2)$$

[1]WLOG we assume the quadrature pair to consist of a purely even and a purely odd RF.

Note that the two models are identical in their monocular inputs and the monocular part of their output (the first four terms in equations 1 and 2) and only vary in their binocular interaction terms (in brackets). The only way in which the first model can implement preferred disparities other than zero is by a positional displacement of the RFs in the two eyes with respect to each other (the disparity tuning curve achieves its maximum when the disparity in the image matches the disparity between the RFs). The second model, on the other hand achieves non-zero preferred disparities by employing a phase shift between the left and right RF (90 deg in our case). It is therefore considered to be phase-disparity model, while the first one is called a position disparity one.[2]

## 3    Results

How much information the response of a neuron carries about a particular stimulus attribute depends both on the sensitivity of the response to changes in that attribute and to the variability (or uncertainty) in the response across all stimuli while keeping that attribute fixed. Fisher information is the standard way to quantify this intuition in the context of intrinsic noise ([6], but also see [2]) and we will use it to evaluate the binocular energy model mechanisms with regard to their ability to extract the disparity information contained in the monocular inputs arriving at the eyes.

### 3.1    Response variability

Figure 2 shows the mean of the binocular response of the two models. The variation of the response around the mean due to the variation in monocular image structure in the RDS is shown in Figure 3 (top row) for four exemplary disparities: $-1, 0, 1$ and uncorrelated ($\pm\infty$), indicated in Figure 2. Unlike in the commonly assumed case of intrinsic noise, $p_{\text{binoc}}(r|d)$ – the stimulus-conditioned response distribution – is far from Poisson or Gaussian. Interestingly, its mode is always at zero – the average response to uncorrelated stimuli – and the fact that the mean depends on the stimulus disparity is primarily due to the disparity-dependence of the skew of the response distribution (Figure 3).[3] The skew in turn depends on the disparity through the disparity-dependent correlation between the RF outputs as illustrated in Figure 3 (bottom row). Of particular interest are the response distributions

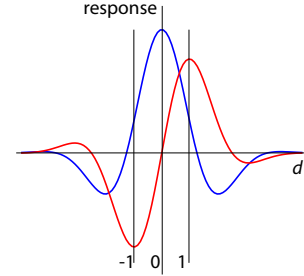

Figure 2: Binocular responses for even (blue) and odd (red) model.

at the zero disparity [4], the disparities at which $r_{\text{odd}}$ takes its minimum and maximum, respectively, and the uncorrelated case (infinite disparity). In the case of infinite disparity, the images in the two eyes are completely independent of each other and hence the outputs of the left and right RFs are independent Gaussians. Therefore, $\nu_{\text{L}}\nu_{\text{R}} \sim p_{\text{binoc}}(r|d=\infty)$ is symmetric around 0. In the case of zero disparity (identical images in left and right eye), the correlation is 1 between the outputs of left and right RFs (both even, or both odd). It follows that $\nu_{\text{L}}\nu_{\text{R}} \sim \chi_1^2$ and hence has a mean of 1. What is also apparent is that the binocular energy model with phase disparity (where each even-symmetric RF is paired with an odd-symmetric one) never achieves perfect correlation between the left and right eye and only covers smaller values.

### 3.2    Fisher information

#### 3.2.1    Fisher information contained in monocular inputs

First, we quantify the information contained in the inputs to the energy model by using Fisher information. Consider the 4D space spanned by the outputs of the four RFs in left and right eye: $(\nu_{\text{L}}^{\text{e}}, \nu_{\text{L}}^{\text{o}}, \nu_{\text{R}}^{\text{e}}, \nu_{\text{R}}^{\text{o}})$. Since the $\nu$ are drawn from identical Gaussians[5], the mean responses of the

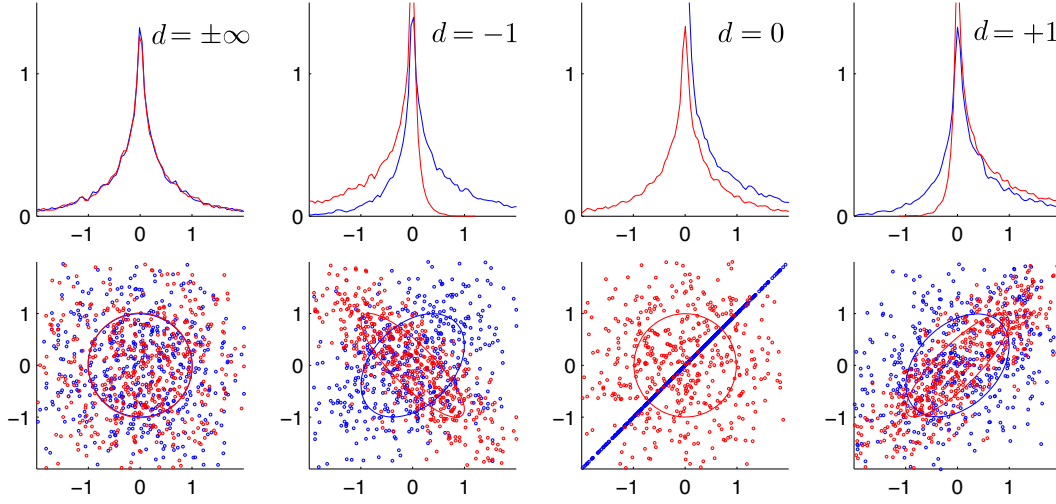

Figure 3: Response distributions $p(r|d)$ for varying $d$. **Top row:** histograms for values of interaction terms $\nu_L^e \nu_R^e$ (blue) and $\nu_L^e \nu_R^o$ (red). **Bottom row:** distribution of corresponding RF outputs $\nu_L$ vs $\nu_R$. $1\sigma$ curves are shown to indicate correlations. Blue ($\nu_L^e$ vs $\nu_R^e$) and red ($\nu_L^e$ vs $\nu_R^o$) colors refer to the model with even-symmetric tuning curve and odd-symmetric tuning curve, respectively. The disparity value for each column is $\pm\infty, -1, 0$ and $1$ corresponding to those highlighted in Figure 2.

monocular inputs do not depend on the stimulus and hence, the Fisher information is given by $\mathcal{I}(d) = \frac{1}{2}\text{tr}(C^{-1}C'C^{-1}C')$ where $C$ is the covariance matrix belonging to $(\nu_L^e, \nu_L^o, \nu_R^e, \nu_R^o)$:

$$C = \begin{pmatrix} 1 & 0 & a(d) & c(d) \\ 0 & 1 & c(-d) & a(d) \\ a(d) & c(-d) & 1 & 0 \\ c(d) & a(d) & 0 & 1 \end{pmatrix}$$

where we model the interaction terms $a(d) := \langle \nu_L^e \nu_R^e \rangle = \langle \nu_L^o \nu_R^o \rangle$ and $c(d) := \langle \nu_L^e \nu_R^o \rangle$ as Gabor functions[6] since Gabors functions have been shown to provide a good fit to the range of RF shapes and disparity tuning curves that are empirically observed in early sensory cortex ([5]).[7] $a(d)$ and $c(d)$ are illustrated by the blue and red curves in Figure 2, respectively. Because the binocular part of the energy model response, or disparity tuning curve, is the convolution of the left and right RFs, the phase of the Gabor describing the disparity tuning curve is given by the difference between the phases of the corresponding RFs. Therefore $c(d)$ is odd-symmetric and $c(-d) = -c(d)$. We obtain

$$\mathcal{I}_{\text{inputs}}(d) = \frac{2}{(1-a^2-c^2)^2} \left[ (1+a^2-c^2)a'^2 + (1+c^2-a^2)c'^2 + 4aca'c' \right] \qquad (3)$$

where we omitted the stimulus dependence of $a(d)$ and $c(d)$ for clarity of exposition and where $'$ denotes the 1st derivative with respect to the stimulus $d$. The denominator of equation (3)) is given by $\det C$ and corresponds to the Gaussian envelope of the Gabor functions for $a(d)$ and $c(d)$:

$$\det C = 1 - a^2 - c^2 = 1 - \exp(-\frac{s^2}{\sigma^2}).$$

In Figure 4B (black) we plot the Fisher information as a function of disparity. We find that the Fisher information available in the inputs diverges at zero disparity (at the difference between the centers of the left and right RFs in general). This means that the ability to discriminate zero disparity from

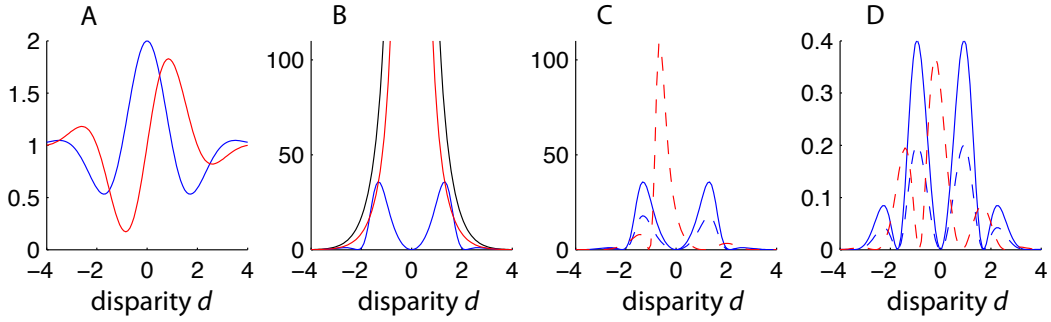

Figure 4: **A**: Disparity tuning curves for the model using position disparity (even) and phase disparity (odd) in blue and red, respectively. **B**: Black: Fisher information contained in the monocular inputs. Blue: Fisher information left after combining inputs from left and right eye according to position disparity model. Red: Fisher information after combining inputs using phase disparity model. Note that the black and red curves diverge at zero disparity. **C**: Fisher information for the final model output/neuronal response. Same color code as previously. Solid lines correspond to complex, dashed lines to simple cells. **D**: Same as C but with added Gaussian noise in the monocular inputs.

nearby disparities is arbitrarily good. In reality, intrinsic neuronal variability will limit the Fisher information at zero.[8]

### 3.2.2 Combination of left and right inputs

Next we analyze the information that remains after linearly combining the monocular inputs in the energy model. It follows that the 4-dimensional monocular input space is reduced to a 2-dimensional binocular one for each model, sampled by $(\nu_L^e + \nu_R^e, \nu_L^o + \nu_R^o)$ and $(\nu_L^e + \nu_R^o, \nu_L^o + \nu_R^e)$, respectively. Again, the marginal distributions are Gaussians with zero mean independent of stimulus disparity. This means that we can compute the Fisher information for the position disparity model from the covariance matrix $C$ as above:

$$
\begin{aligned}
C_{\text{even}} &= \begin{pmatrix} \langle(\nu_L^e + \nu_R^e)^2\rangle & \langle(\nu_L^e + \nu_R^e)(\nu_L^o + \nu_R^o)\rangle \\ \langle(\nu_L^e + \nu_R^e)(\nu_L^o + \nu_R^o)\rangle & \langle(\nu_L^o + \nu_R^o)^2\rangle \end{pmatrix} \\
&= \begin{pmatrix} 2 + 2a & 0 \\ 0 & 2 + 2a \end{pmatrix}
\end{aligned}
$$

Here we exploited that $\langle \nu_L^e \nu_L^o \rangle = \langle \nu_R^e \nu_R^o \rangle = 0$ since the even and odd RFs are orthogonal and that $\langle \nu_L^e \nu_R^o \rangle = -\langle \nu_R^o \nu_L^e \rangle$. The Fisher information follows as

$$
\mathcal{I}_{\text{even}}(d) = \frac{a'(d)^2}{[1 + a(d)]^2}. \tag{4}
$$

The dependence of Fisher information on $d$ is shown in Figure 4B (blue). The total information (as measured by integrating Fisher information over all disparities) communicated by the position-disparity model is greatly reduced compared to the total Fisher information present in the inputs. $a(d)$ is an even-symmetric Gabor (illustrated in Figure 2) and hence the Fisher information is greatest on either side of the maximum where the slopes of $a(d)$ are steepest, and zero at the center where $a(d)$ has its peak. We note here that the Fisher information for the final tuning curve for the position-disparity model is the same as in equation (4) and therefore we will postpone a more detailed discussion of it until section 3.2.3.

On the other hand, when combining the monocular inputs according to the phase disparity model, we find:

$$C_{\mathrm{odd}} = \begin{pmatrix} \langle (\nu_{\mathrm{L}}^{\mathrm{e}} + \nu_{\mathrm{R}}^{\mathrm{o}})^2 \rangle & \langle (\nu_{\mathrm{L}}^{\mathrm{e}} + \nu_{\mathrm{R}}^{\mathrm{o}})(\nu_{\mathrm{L}}^{\mathrm{o}} + \nu_{\mathrm{R}}^{\mathrm{e}}) \rangle \\ \langle (\nu_{\mathrm{L}}^{\mathrm{e}} + \nu_{\mathrm{R}}^{\mathrm{o}})(\nu_{\mathrm{L}}^{\mathrm{o}} + \nu_{\mathrm{R}}^{\mathrm{e}}) \rangle & \langle (\nu_{\mathrm{L}}^{\mathrm{o}} + \nu_{\mathrm{R}}^{\mathrm{e}})^2 \rangle \end{pmatrix}$$

$$= \begin{pmatrix} 2 + 2c & 2a \\ 2a & 2 - 2c \end{pmatrix}$$

since again $\langle \nu_{\mathrm{L}}^{\mathrm{e}} \nu_{\mathrm{L}}^{\mathrm{o}} \rangle = \langle \nu_{\mathrm{R}}^{\mathrm{e}} \nu_{\mathrm{R}}^{\mathrm{o}} \rangle = 0$ and $\langle \nu_{\mathrm{L}}^{\mathrm{e}} \nu_{\mathrm{R}}^{\mathrm{o}} \rangle = -\langle \nu_{\mathrm{R}}^{\mathrm{o}} \nu_{\mathrm{L}}^{\mathrm{e}} \rangle = c$. The Fisher information in this case follows as

$$\mathcal{I}_{\mathrm{odd}}(d) = \frac{1}{(1 - a^2 - c^2)^2} \left[ (1 + a^2 - c^2)a'^2 + (1 + c^2 - a^2)c'^2 + 4aca'c' \right]$$

$$= \frac{1}{2} \mathcal{I}_{\mathrm{inputs}}(d)$$

$\mathcal{I}_{\mathrm{odd}}(d)$ is shown in Figure 4B (red). While loosing 50% of the Fisher information present in the inputs, the Fisher information after combining left and right RF outputs is much larger in this case than for the position disparity model explored above. How can that be? Why are the two ways of combining the monocular outputs not symmetric? Insight into this question can be gained by looking at the binocular interaction terms in the quadratic expansion of the feature space for the two models.[9] For the position disparity model we obtain the 3-dimensional space $(\nu_{\mathrm{L}}^{\mathrm{e}} \nu_{\mathrm{R}}^{\mathrm{e}}, \nu_{\mathrm{L}}^{\mathrm{o}} \nu_{\mathrm{R}}^{\mathrm{o}}, \nu_{\mathrm{L}}^{\mathrm{e}} \nu_{\mathrm{R}}^{\mathrm{o}} + \nu_{\mathrm{L}}^{\mathrm{o}} \nu_{\mathrm{R}}^{\mathrm{e}})$ of which the third dimension cannot contribute to the Fisher information since $\nu_{\mathrm{L}}^{\mathrm{e}} \nu_{\mathrm{R}}^{\mathrm{o}} + \nu_{\mathrm{L}}^{\mathrm{o}} \nu_{\mathrm{R}}^{\mathrm{e}} = 0$. In the phase-disparity model, however, the quadratic expansion yields $(\nu_{\mathrm{L}}^{\mathrm{e}} \nu_{\mathrm{R}}^{\mathrm{o}}, \nu_{\mathrm{L}}^{\mathrm{o}} \nu_{\mathrm{R}}^{\mathrm{e}}, \nu_{\mathrm{L}}^{\mathrm{e}} \nu_{\mathrm{R}}^{\mathrm{e}} + \nu_{\mathrm{L}}^{\mathrm{o}} \nu_{\mathrm{R}}^{\mathrm{o}})$. Here, all three dimensions are linearly independent (although correlated), each contributing to the Fisher information. This can also explain why $\mathcal{I}_{\mathrm{odd}}(d)$ is symmetric around zero, and independent of the Gabor phase of $c(d)$. While this is not a rigorous analysis yet of the differences between the models at the stage of binocular combination, it serves as a starting point for a future investigation.

### 3.2.3 Disparity tuning curves

In order to collapse the 2-dimensional binocular inputs into a scalar output that can be coded in the spike rate of a neuron, the energy model postulates a squaring output nonlinearity after each linear combination and summing the results. Since the $(\nu_{\mathrm{L}} + \nu_{\mathrm{R}})^2$ are not Normally distributed and their means depend on the stimulus disparity, we cannot employ the above approach to calculate Fisher information but instead use the more general

$$\mathcal{I}(d) = \mathrm{E}\left[ \left( \frac{\partial}{\partial d} \ln p(r; d) \right)^2 \right] = \int_0^\infty p(r; d) \left( \frac{\partial}{\partial d} \ln p(r; d) \right)^2 \, \mathrm{d}r \qquad (5)$$

where $p(r; d)$ is the response distribution for stimulus disparity $d$. Because the $\nu$ are drawn from a Gaussian with variance 1, $\nu_{\mathrm{L}}^{\mathrm{e}} + \nu_{\mathrm{R}}^{\mathrm{e}}$ and $\nu_{\mathrm{L}}^{\mathrm{o}} + \nu_{\mathrm{R}}^{\mathrm{o}}$ are drawn from $\mathcal{N}[0, 2(1 + a(d))]$ since we defined $a(d) = \langle \nu_{\mathrm{L}}^{\mathrm{e}} \nu_{\mathrm{R}}^{\mathrm{e}} \rangle = \langle \nu_{\mathrm{L}}^{\mathrm{o}} \nu_{\mathrm{R}}^{\mathrm{o}} \rangle$. Conditioned on $d$, $(\nu_{\mathrm{L}}^{\mathrm{e}} + \nu_{\mathrm{R}}^{\mathrm{e}})^2$ and $(\nu_{\mathrm{L}}^{\mathrm{o}} + \nu_{\mathrm{R}}^{\mathrm{o}})^2$ are independent and it follows for the model with an even-symmetric tuning function that

$$\frac{1}{2[1 + a(d)]} \left[ (\nu_{\mathrm{L}}^{\mathrm{e}} + \nu_{\mathrm{R}}^{\mathrm{e}})^2 + (\nu_{\mathrm{L}}^{\mathrm{o}} + \nu_{\mathrm{R}}^{\mathrm{o}})^2 \right] \sim \chi_2^2 \quad \text{and}$$

$$p_{\mathrm{even}}(r; d) = \frac{1}{4[1 + a(d)]} \exp\left\{ -\frac{r}{4[1 + a(d)]} \right\} H(r) \qquad (6)$$

where $H(r)$ is the Heaviside step function.[10] Substituting equation (6) into equation (5) we find[11]

$$\mathcal{I}_{\mathrm{even}}^{\mathrm{complex}}(d) = \frac{a'(d)^2}{4[1 + a(d)]^3} \int_0^\infty \mathrm{d}r \left[ \frac{r}{4[1 + a(d)]} - 1 \right]^2 \exp\left\{ -\frac{r}{4[1 + a(d)]} \right\}$$

$$= \frac{a'(d)^2}{[1 + a(d)]^2} \qquad (7)$$

Remarkably, this is exactly the same amount of information that is available after summing left and right RFs (see equation 4), so none is lost after squaring and combining the quadrature pair. We show $\mathcal{I}_{\text{even}}(d)$ in Figure 4C (blue). It is also interesting to note that the general form for $\mathcal{I}_{\text{even}}(d)$ differs from the Fisher information based on the Poisson noise model (and ignoring stimulus variability as considered here) only by the exponent of 2 in the denominator. Since $1 + a(d) \geq 0$ this means that the qualitative dependence of $\mathcal{I}$ on $d$ is the same, the main difference being that the Fisher information favors small over large spike rates even more. Conversely, it follows that when Fisher information only takes the neuronal noise into consideration, it greatly overestimates the information that the neuron carries with respect to the to-be-inferred stimulus parameter for realistic spike counts (of greater than two). Furthermore, unlike in the Poisson case, a scaling up of the tuning function $1 + a(d)$ does not translate into greater Fisher information. Fisher information with respect to stimulus variability as considered here is invariant to the absolute height of the tuning curve.[12]

Considering the phase-disparity model, $(\nu_{\text{L}}^{\text{e}} + \nu_{\text{R}}^{\text{o}})^2$ and $(\nu_{\text{L}}^{\text{o}} + \nu_{\text{R}}^{\text{e}})^2$ are drawn from $\mathcal{N}[0, 2(1+c(d))]$ and $\mathcal{N}[0, 2(1+c(d))]$, respectively, since $c(d) = \langle \nu_{\text{L}}^{\text{e}} \nu_{\text{R}}^{\text{o}} \rangle = -\langle \nu_{\text{L}}^{\text{o}} \nu_{\text{R}}^{\text{e}} \rangle$. Unfortunately, since $\nu_{\text{L}}^{\text{o}} + \nu_{\text{R}}^{\text{o}}$ and $\nu_{\text{L}}^{\text{o}} + \nu_{\text{R}}^{\text{e}}$ have different variances depending on $d$, and are usually not independent of each other, the sum cannot be modeled by a $\chi^2-$distribution. However, we can compute the Fisher information for the two implied binocular simple cells instead.[13] It follows that

$$\frac{1}{2[1 + c(d)]} \left[ (\nu_{\text{L}}^{\text{e}} + \nu_{\text{R}}^{\text{o}})^2 \right] \quad \sim \quad \chi_1^2 \ \text{ and}$$

$$p_{\text{odd}}^{\text{simple}}(r; d) \quad = \quad \frac{1}{2\Gamma(1/2)\sqrt{1 + c(d)}} \frac{1}{\sqrt{r}} \exp\left\{ -\frac{r}{4[1 + c(d)]} \right\} H(r).$$

and[14]

$$\mathcal{I}_{\text{odd}}^{\text{simple}}(d) \quad = \quad \frac{1}{2\Gamma(1/2)} \frac{c'(d)^2}{\sqrt{1 + c(d)}^5} \int_0^{\infty} \mathrm{d}r \, \frac{1}{\sqrt{r}} \left[ \frac{r}{4[1 + c(d)]} - \frac{1}{2} \right]^2 \exp\left\{ -\frac{r}{4[1 + c(d)]} \right\}$$

$$= \quad \frac{1}{2} \frac{c'(d)^2}{[1 + c(d)]^2}$$

[15] The dependence of $\mathcal{I}_{\text{odd}}^{\text{simple}}$ on disparity is shown in Figure 4C (red dashed). Most of the Fisher information is located in the primary slope (compare Figure 4A) followed by secondary slope to its left. The reason for this is the strong boost Fisher information gets when responses are lowest. We also see that the total Fisher information carried by a phase-disparity simple cell is significantly higher than that carried by a position-disparity simple cell (compare dashed red and blue lines) raising the question of what other advantages or trade-offs there are that make it beneficial for the primate brain to employ so many position-disparity ones. Intrinsic neuronal variability may provide part of the answer since the difference in Fisher information between both models decreases as intrinsic variability increases. Figure 4D shows the Fisher information after Gaussian noise has been added to the monocular inputs. However, even in this high intrinsic noise regime (noise variance of the same order as tuning curve amplitude) the model with phase disparity carries significantly more total Fisher information.

# 4 Discussion

The central idea of our paper is to evaluate the quality of a sensory code with respect to an inference task by taking stimulus variability into account, in particular that induced by irrelevant stimulus attributes. By framing stimulus-induced nuisance variability as noise, we were able to employ the existing framework of Fisher information for evaluating the standard model of early binocular processing with respect to inferring disparity from random dot stereograms.

We started by investigating the disparity-conditioned variability of the binocular response in the absence of intrinsic neuronal noise. We found that the response distributions are far from Poisson or Gaussian and – independent of stimulus disparity – are always peaked at zero (the mean response to uncorrelated images). The information contained in the correlations between left and right RF outputs are translated into a modulation of the neuron's mean firing rate primarily by altering the skew of the response distribution. This is quite different from the case of intrinsic noise and has implications for comparing different codes. It is noteworthy that these response distributions are entirely imposed by the sensory system – the combination of the structure of the external world with the internal processing model. Unlike the case of intrinsic noise which is usually added ad-hoc *after* the neuronal computation has been performed, in our case the computational model impacts the usefulness of the code beyond the traditionally reported tuning functions. This property extends to the case of population codes, the next step for future work. Of great importance for the performance of population codes are interneuronal correlations. Again, the noise correlations due to nuisance stimulus parameters are a direct consequence of the processing model and the structure of the external input.

Next we compared the Fisher information available for our inference task at various stages of binocular processing. We computed the Fisher information available in the monocular inputs to binocular neurons in V1, after binocular combination and after the squaring nonlinearity required to translate binocular correlations into mean firing rate modulation. We find that despite the great stimulus variability, the total Fisher information available in the inputs diverges and is only bounded by intrinsic neuronal variability. The same is still true after binocular combination for one flavor of the model considered here – that employing phase disparity (or pairing unlike RFs in either eye), not the other one (position disparity), which has lost most information after the initial combination. At this point, our new approach allows us to ask a normative question: In what way should the monocular inputs be combined so as to lose a minimal amount of information about the relevant stimulus dimension? Is the combination proposed by the standard model to obtain even-symmetric tuning curves the only one to do so or are they others that produce a different tuning curve, with a different response distribution that is more suited to inferring depth? Conversely, we can compare our results for the model stages leading from simple to complex cells and compare them with the corresponding Fisher information computed from empirically observed distributions, to test our model assumptions.

Recently, Fisher information has been criticized as a tool for comparing population codes ([3, 2]). We note that our approach can be readily adapted to other measures like mutual information or their framework of neurometric function analysis to compare the performance of different codes in a disparity discrimination task.

Another potentially promising avenue of future research would to investigate the effect of thresholding on inference performance. One reason that odd-symmetric tuning curves had higher Fisher information in the case we investigated was that odd-symmetric cells produce near-zero responses more often in the context of the energy model. However, it is known from empirical observations that fitting even-symmetric disparity tuning curves requires an additional thresholding output nonlinearity. It is unclear at this point to what extend such a change to the response distribution helps or hinders inference.

And finally, we suggest that considering the different shapes of response distributions induced by the specifics of the sensory modality might have an impact on the discussion about probabilistic population codes ([7, 8] and references therein). Cue-integration, for instance, has usually been studied under the assumption of Poisson-like response distributions, assumptions that do not appear to hold in the case of combining disparity cues from different parts of the visual field.

**Acknowledgments**

This work has been supported by the Bernstein award to MB (BMBF; FKZ: 01GQ0601).

## Footnotes

[2]We use position disparity model and even-symmetric tuning interchangeably, as well as phase disparity model and odd-symmetric tuning. Unfortunately, the term disparity is used for both disparities between the RFs, and for disparities between left and right images (in the stimulus). If not indicated otherwise, we will always refer to stimulus disparity for the rest of the paper.

[3]The RF outputs are Normally distributed in the limit of infinitely many dots (RFs act as linear filters + central limit theorem). Therefore the disparity-conditioned responses $p(r|d)$ correspond to the off-diagonal terms in a Wishart distribution, marginalized over all the other matrix elements.

[4]WLOG we assume the displacement between the RF centers in the left and right eye to be zero.

[5]The model RFs have been normalized by their variance, such that $\text{var}[\nu] = 1$ and $\nu \sim \mathcal{N}(0,1)$.

[6]A Gabor function is defined as $\cos(2\pi f d - \phi) \exp[-\frac{(d-d_0)^2}{2\sigma^2}]$ were $f$ is spatial frequency, $d$ is disparity, $\phi$ is the Gabor phase, $d_o$ is the envelope center (set to zero here, WLOG) and $\sigma$ the envelope bandwidth.

[7]The assumption that the binocular interaction can be modeled by a Gabor is not important for the principal results of this paper. In fact, the formulas for the Fisher information in the monocular inputs and in the disparity tuning curves derived below hold for other (reasonable) choices for $a(d)$ and $c(d)$ as well.

[8]E.g. additive Gaussian noise with variance $\sigma^{N2}$ on the monocular filter outputs eliminates the singularity: $\det C = 1 + \sigma^{N2} - a^2 - c^2 \geq \sigma^{N2}$.

[9]By quadratic expansion of the feature space we refer to expanding a 2-dimensional feature space $(f_1, f_2)$ to a 3-dimensional one $(f_1^2, f_2^2, f_1 f_2)$ by considering the binocular interaction terms in all quadratic forms.

[10]We see that $\langle r \rangle_{p_{\mathrm{even}}(r; d)} = 4[1 + a(d)]$ and hence we recover the Gabor-shaped tuning function that we introduced in section 3.2.1 to model the empirically observed relationship between disparity $d$ and mean spike rate $r$.

[11]$\int_0^\infty \mathrm{d}x \, (x/\alpha - 1)^2 \exp(-x/\alpha) = \alpha$ for $\alpha > 0$.

[12]What is outside of the scope of this paper but obvious from equation (7) is that Fisher information is maximized when the denominator, or the tuning function is minimal. Within the context of the energy model, this occurs for neither the position-disparity model, nor the classic phase-disparity one, but for a model where the left and right RFs that are linearly combined, are inverted with respect to each other (i.e. phase-shifted by $\pi$). In that case $a(d)$ is a Gabor function with phase $\pi$ and becomes zero at zero disparity such that the Fisher information diverges. Such neurons, called tuned-inhibitory (TI, [11]) make up a small minority of neurons in monkey V1.

[13]The energy model as presented thus far models the responses of binocular complex cells. Disparity-selective simple cells are typically modeled by just one combination of left and right RFs $(\nu_{\text{L}}^{\text{e}} + \nu_{\text{R}}^{\text{o}})^2$ or $(\nu_{\text{L}}^{\text{o}} + \nu_{\text{R}}^{\text{e}})^2$, and not the entire quadrature pair.

[14]$\int_0^{\infty} \mathrm{d}x \, \sqrt{x}^{-1} (x/\alpha - 1/2)^2 \exp(-x/\alpha) = \sqrt{\pi}\sqrt{\alpha}/2$ for $\alpha > 0$.

[15]This derivation equally applies to the Fisher information of simple cells with position disparity by substituting $a(d)$ for $c(d)$ and we obtain $\mathcal{I}_{\text{even}}^{\text{simple}}(d) = \frac{1}{2} \frac{a'(d)^2}{[1+a(d)]^2}$. This function is shown in Figure 4C (blue dashed).

# References

[1] LF Abbott and P Dayan. The effect of correlated variability on the accuracy of a population code. *Neural Comput*, 11(1):91–101, 1999.

[2] P Berens, S Gerwinn, A Ecker, and M Bethge. Neurometric function analysis of population codes. In Y. Bengio, D. Schuurmans, J. Lafferty, C. K. I. Williams, and A. Culotta, editors, *Advances in Neural Information Processing Systems 22*, pages 90–98. 2009.

[3] M Bethge, D Rotermund, and K Pawelzik. Optimal short-term population coding: when fisher information fails. *Neural Comput*, 14(10):2317–2351, 2002.

[4] C Blakemore and B Julesz. Stereoscopic depth aftereffect produced without monocular cues. *Science*, 171(968):286–288, 1971.

[5] BG Cumming and GC DeAngelis. The physiology of stereopsis. *Annu Rev Neurosci*, 24:203–238, 2001.

[6] P Dayan and LF Abbott. *Theoretical neuroscience: Computational and mathematical modeling of neural systems*. MIT Press, 2001.

[7] J Fiser, P Berkes, G Orban, and M Lengyel. Statistically optimal perception and learning: from behavior to neural representations. *Trends Cogn Sci*, 14(3):119–130, 2010.

[8] WJ Ma, JM Beck, PE Latham, and A Pouget. Bayesian inference with probabilistic population codes. *Nat Neurosci*, 9(11):1432–1438, 2006.

[9] David Marr. *Vision: A Computational Investigation into the Human Representation and Processing of Visual Information*. Henry Holt and Co., Inc., New York, NY, USA, 1982.

[10] I Ohzawa, GC DeAngelis, and RD Freeman. Stereoscopic depth discrimination in the visual cortex: neurons ideally suited as disparity detectors. *Science*, 249(4972):1037–1041, 1990.

[11] GF Poggio and B Fischer. Binocular interaction and depth sensitivity in striate and prestriate cortex of behaving rhesus monkey. *J Neurophysiol*, 40(6):1392–1405, 1977.

[12] F. Rieke, D. Warland, R.R. van, Steveninck, and W. Bialek. *Spikes: exploring the neural code*. MIT Press, Cambridge, MA, 1997.

